# How to Dynamically Merge Markov Decision Processes

**Satinder Singh**
Department of Computer Science
University of Colorado
Boulder, CO 80309-0430
baveja@cs.colorado.edu

**David Cohn**
Adaptive Systems Group
Harlequin, Inc.
Menlo Park, CA 94025
cohn@harlequin.com

## Abstract

We are frequently called upon to perform multiple tasks that compete for our attention and resource. Often we know the optimal solution to each task in isolation; in this paper, we describe how this knowledge can be exploited to efficiently find good solutions for doing the tasks in parallel. We formulate this problem as that of dynamically merging multiple Markov decision processes (MDPs) into a composite MDP, and present a new theoretically-sound dynamic programming algorithm for finding an optimal policy for the composite MDP. We analyze various aspects of our algorithm and illustrate its use on a simple merging problem.

Every day, we are faced with the problem of doing multiple tasks in parallel, each of which competes for our attention and resource. If we are running a job shop, we must decide which machines to allocate to which jobs, and in what order, so that no jobs miss their deadlines. If we are a mail delivery robot, we must find the intended recipients of the mail while simultaneously avoiding fixed obstacles (such as walls) and mobile obstacles (such as people), and still manage to keep ourselves sufficiently charged up.

Frequently we know how to perform each task in isolation; this paper considers how we can take the information we have about the individual tasks and combine it to efficiently find an optimal solution for doing the entire set of tasks in parallel. More importantly, we describe a theoretically-sound algorithm for doing this merging dynamically; new tasks (such as a new job arrival at a job shop) can be assimilated online into the solution being found for the ongoing set of simultaneous tasks.

# 1    The Merging Framework

Many decision-making tasks in control and operations research are naturally formulated as Markov decision processes (MDPs) (e.g., Bertsekas & Tsitsiklis, 1996). Here we define MDPs and then formulate what it means to have multiple simultanous MDPs.

## 1.1    Markov decision processes (MDPs)

An MDP is defined via its state set $S$, action set $A$, transition probability matrices $P$, and payoff matrices $R$. On executing action $a$ in state $s$ the probability of transiting to state $s'$ is denoted $P^a(ss')$ and the expected payoff associated with that transition is denoted $R^a(ss')$. We assume throughout that the payoffs are non-negative for all transitions. A policy assigns an action to each state of the MDP. The value of a state under a policy is the expected value of the discounted sum of payoffs obtained when the policy is followed on starting in that state. The objective is to find an optimal policy, one that maximizes the value of every state. The optimal value of state $s$, $V^*(s)$, is its value under the optimal policy.

The optimal value function is the solution to the Bellman optimality equations: for all $s \in S$, $V(s) = \max_{a \in A}(\sum_{s'} P^a(ss')[R^a(ss')+\gamma V(s')])$, where the discount factor $0 \leq \gamma < 1$ makes future payoffs less valuable than more immediate payoffs (e.g., Bertsekas & Tsitsiklis, 1996). It is known that the optimal policy $\pi^*$ can be determined from $V^*$ as follows: $\pi^*(s) = \text{argmax}_{a \in A}(\sum_{s'} P^a(ss')[R^a(ss') + \gamma V^*(s')])$. Therefore solving an MDP is tantamount to computing its optimal value function.

## 1.2    Solving MDPs via Value Iteration

Given a model $(S, A, P, R)$ of an MDP value iteration (e.g., Bertsekas & Tsitsiklis, 1996) can be used to determine the optimal value function. Starting with an initial guess, $V_0$, iterate for all $s$ $V_{k+1}(s) = \max_{a \in A}(\sum_{s' \in S} P^a(ss')[R^a(ss') + \gamma V_k(s')])$. It is known that $\max_{s \in S} |V_{k+1}(s) - V^*(s)| \leq \gamma \max_{s \in S} |V_k(s) - V^*(s)|$ and therefore $V_k$ converges to $V^*$ as $k$ goes to infinity. Note that a Q-value (Watkins, 1989) based version of value iteration and our algorithm presented below is also easily defined.

## 1.3    Multiple Simultaneous MDPs

The notion of an optimal policy is well defined for a single task represented as an MDP. If, however, we have multiple tasks to do in parallel, each with its own state, action, transition probability, and payoff spaces, optimal behavior is not automatically defined. We will assume that payoffs sum across the MDPs, which means we want to select actions for each MDP at every time step so as to maximize the expected discounted value of this summed payoff over time. If actions can be chosen independently for each MDP, then the solution to this "composite" MDP is obvious — do what's optimal for each MDP. More typically, choosing an action for one MDP constrains what actions can be chosen for the others. In a job shop for example, actions correspond to assignment of resources, and the same physical resource may not be assigned to more than one job simultaneously.

Formally, we can define a composite MDP as a set of $N$ MDPs $\{M^i\}_1^N$. We will use superscripts to distinguish the component MDPs, e.g., $S^i$, $A^i$, $P^i$, and $R^i$ are the state, action, transition probability and payoff parameters of MDP $M^i$. The state space of the composite MDP, $S$, is the cross product of the state spaces of the component MDPs, i.e., $S = S^1 \times S^2 \times \ldots \times S^N$. The constraints on actions implies that

the action set of the composite MDP, $A$, is some proper subset of the cross product of the $N$ component action spaces. The transition probabilities and the payoffs of the composite MDP are *factorial* because the following decompositions hold: for all $s, s' \in S$ and $a \in A$, $P^a(ss') = \Pi_{i=1}^N P^{a^i}(s^i s^{i'})$ and $R^a(ss') = \sum_{i=1}^N R^{a^i}(s^i s^{i'})$. Singh (1997) has previously studied such factorial MDPs but only for the case of a fixed set of components.

The optimal value function of a composite MDP is well defined, and satisfies the following Bellman equation: for all $s \in S$,

$$V(s) = \max_{a \in A} \sum_{s' \in S} \left( \Pi_{i=1}^N P^{a^i}(s^i s^{i'}) \Big[ \sum_{i=1}^N R^{a^i}(s^i s^{i'}) + \gamma V(s') \Big] \right). \tag{1}$$

Note that the Bellman equation for a composite MDP assumes an identical discount factor across component MDPs and is not defined otherwise.

## 1.4 The Dynamic Merging Problem

Given a composite MDP, and the optimal solution (e.g. the optimal value function) for each of its component MDPs, we would like to efficiently compute the optimal solution for the composite MDP. More generally, we would like to compute the optimal composite policy given only *bounds* on the value functions of the component MDPs (the motivation for this more general version will become clear in the next section). To the best of our knowledge, the dynamic merging question has not been studied before.

Note that the traditional treatment of problems such as job-shop scheduling would formulate them as nonstationary MDPs (however, see Zhang and Dietterich, 1995 for another learning approach). This normally requires augmenting the state space to include a "time" component which indexes all possible state spaces that could arise (e.g., Bertsekas, 1995). This is inefficient, and potentially infeasible unless we know in advance all combinations of possible tasks we will be required to solve. One contribution of this paper is the observation that this type of nonstationary problem can be reformulated as one of dynamically merging (individually) stationary MDPs.

### 1.4.1 The naive greedy policy is suboptimal

Given bounds on the value functions of the component MDPs, one heuristic composite policy is that of selecting actions according to a one-step greedy rule:

$$\pi(s) = \underset{a}{\text{argmax}} (\sum_{s'} \Pi_{i=1}^N P^{a^i}(s^i s^{i'}) [\sum_{i=1}^N (R^{a^i}(s^i, a^i) + \gamma X^i(s^{i'}))]),$$

where $X^i$ is the upper or lower bound of the value function, or the mean of the bounds. It is fairly easy however, to demonstrate that these policies are substantially suboptimal in many common situations (see Section 3).

## 2 Dynamic Merging Algorithm

Consider merging $N$ MDPs; job-shop scheduling presents a special case of merging a new single MDP with an old composite MDP consisting of several factor MDPs. One obvious approach to finding the optimal composite policy would be to directly perform value iteration in the composite state and action space. A more efficient approach would make use of the solutions (bounds on optimal value functions) of the existing components; below we describe an algorithm for doing this.

Our algorithm will assume that we know the optimal values, or more generally, upper and lower bounds to the optimal values of the states in each component MDP. We use the symbols $L$ and $U$ for the lower and upper bounds; if the optimal value function for the $i^{th}$ factor MDP is available then $L^i = U^i = V^{*,i}$.[1]

Our algorithm uses the bounds for the component MDPs to compute bounds on the values of composite states as needed and then incrementally updates and narrows these initial bounds using a form of value iteration that allows pruning of actions that are not *competitive*, that is, actions whose bounded values are strictly dominated by the bounded value of some other action.

**Initial State:** The initial composite state $s_0$ is composed from the start state of all the factor MDPs. In practice (e.g. in job-shop scheduling) the initial composite state is composed of the start state of the new job and whatever the current state of the set of old jobs is. Our algorithm exploits the initial state by only updating states that can occur from the initial state under competitive actions.

**Initial Value Step:** When we need the value of a composite state $s$ for the first time, we compute upper and lower bounds to its optimal value as follows: $L(s) = \max_{i=1}^{N} L^i(s^i)$, and $U(s) = \sum_{i=1}^{N} U^i(s)$.

**Initial Update Step:** We dynamically allocate upper and lower bound storage space for composite states as we first update them. We also create the initial set of competitive actions for $s$ when we first update its value as $A(s) = A$. As successive backups narrow the upper and lower bounds of successor states, some actions will no longer be competitive, and will be eliminated from further consideration.

**Modified Value Iteration Algorithm:**

At step $t$ if the state to be updated is $s_t$:

$$L_{t+1}(s_t) = \max_{a \in A_t(s_t)} \left( \sum_{s'} P^a(s_t s')[R^a(s_t, s') + \gamma L_t(s')] \right)$$

$$U_{t+1}(s_t) = \max_{a \in A_t(s_t)} \left( \sum_{s'} P^a(s_t s')[R^a(s_t, s') + \gamma U_t(s')] \right)$$

$$A_{t+1}(s_t) = \bigcup a \in A_t(s_t) \text{ AND } \sum_{s'} P^a(s_t s')[R^a(s_t, s') + \gamma U_t(s')]$$

$$\geq \underset{b \in A_t(s_t)}{\operatorname{argmax}} \sum_{s'} P^b(s_t s')[R^b(s_t, s') + \gamma L_t(s')]$$

$$s_{t+1} = \begin{cases} s_0 \text{ if } s^i \text{ is terminal for all } s^i \in s \\ s' \in S \text{ such that } \exists a \in A_{t+1}(s_t), P^a(s_t s') > 0 \text{ otherwise} \end{cases}$$

The algorithm terminates when only one competitive action remains for each state, or when the range of all competitive actions for any state are bounded by an indifference parameter $\epsilon$.

To elaborate, the upper and lower bounds on the value of a composite state are backed up using a form of Equation 1. The set of actions that are considered competitive in that state are culled by eliminating any action whose bounded values is strictly dominated by the bounded value of some other action in $A_t(s_t)$. The next state to be updated is chosen randomly from all the states that have non-zero

probability of occuring from any action in $A_{t+1}(s_t)$ or, if $s_t$ is the terminal state of all component MDPs, then $s_{t+1}$ is the start state again.

A significant advantage of using these bounds is that we can prune actions whose upper bounds are worse than the best lower bound. Only states resulting from remaining competitive actions are backed up. When only one competitive action remains, the optimal policy for that state is known, regardless of whether its upper and lower bounds have converged.

Another important aspect of our algorithm is that it focuses the backups on states that are reachable on currently competitive actions from the start state. The combined effect of only updating states that are reachable from the start state and further only those that are reachable under currently competitive actions can lead to significant computational savings. This is particularly critical in scheduling, where jobs proceed in a more or less feedforward fashion and the composite start state when a new job comes in can eliminate a large portion of the composite state space. Ideas based on Kaelbling's (1990) interval-estimation algorithm and Moore & Atkeson's (1993) prioritized sweeping algorithm could also be combined into our algorithm.

The algorithm has a number of desirable "anytime" characteristics: if we have to pick an action in state $s_0$ before the algorithm has converged (while multiple competitive actions remain), we pick the action with the highest lower bound. If a new MDP arrives before the algorithm converges, it can be accommodated dynamically using whatever lower and upper bounds exist at the time it arrives.

## 2.1   Theoretical Analysis

In this section we analyze various aspects of our algorithm.

**UpperBound Calculation:** For any composite state, the sum of the optimal values of the component states is an upper bound to the optimal value of the composite state, i.e., $V^*(s = s^1, s^2, \ldots, s^N) \leq \sum_{i=1}^N V^{*,i}(s^i)$.

If there were no constraints among the actions of the factor MDPs then $V^*(s)$ would equal $\sum_{i=1}^N V^{*,i}(s^i)$ because of the additive payoffs across MDPs. The presence of constraints implies that the sum is an upper bound. Because $V^{*,i}(s^i) \leq U_t(s^i)$ the result follows.

**LowerBound Calculation:** For any composite state, the maximum of the optimal values of the component states is a lower bound to the optimal value of the composite states, i.e., $V^*(s = s^1, s^2, \ldots, s^N) \geq \max_{i=1}^N V^{*,i}(s^i)$.

To see this for an arbitrary composite state $s$, let the MDP that has the largest component optimal value for state $s$ always choose its component-optimal action first and then assign actions to the other MDPs so as to respect the action constraints encoded in set $A$. This guarantees at least the value promised by that MDP because the payoffs are all non-negative. Because $V^{*,i}(s^i) \geq L_t(s^i)$ the result follows.

**Pruning of Actions:** For any composite state, if the upper bound for any composite action, $a$, is lower than the lower bound for some other composite action, then action $a$ cannot be optimal — action $a$ can then safely be discarded from the max in value iteration. Once discarded from the competitive set, an action never needs to be reconsidered.

Our algorithm maintains the upper and lower bound status of $U$ and $L$ as it updates them. The result follows.

**Convergence:** Given enough time our algorithm converges to the optimal policy and optimal value function for the set of composite states reachable from the start state under the optimal policy.

If every state were updated infinitely often, value iteration converges to the optimal solution for the composite problem independent of the intial guess $V_0$. The difference between standard value iteration and our algorithm is that we discard actions and do not update states not on the path from the start state under the continually pruned competitive actions. The actions we discard in a state are guaranteed not to be optimal and therefore cannot have any effect on the value of that state. Also states that are reachable only under discarded actions are automatically irrelevant to performing optimally from the start state.

## 3   An Example: Avoiding Predators and Eating Food

We illustrate the use of the merging algorithm on a simple avoid-predator-and-eat-food problem, depicted in Figure 1a. The component MDPs are the avoid-predator task and eat-food task; the composite MDP must solve these problems simultaneously. In isolation, the tasks avoid-predator and eat-food are fairly easy to learn. The state space of each task is of size $n^4$; 625 states in the case illustrated. Using value iteration, the optimal solutions to both component tasks can be learned in approximately 1000 backups. Directly solving the composite problem requires $n^6$ states (15625 in our case), and requires roughly 1 million backups to converge.

Figure 1b compares the performance of several solutions to the avoid-predator-and-eat-food task. The opt-predator and opt-food curves shows the performance of value iteration on the two component tasks in isolation; both converge quickly to their optima. While it requires no further backups, the greedy algorithm of Section 1.4.1 falls short of optimal performance. Our merging algorithm, when initialized with solutions for the component tasks (5000 backups each) converges quickly to the optimal solution. Value iteration directly on the composite state space also finds the optimal solutions, but requires 4-5 times as many backups. Note that value iteration in composite state space also updated states on trajectories (as in Barto etal.'s, 1995 RTDP algorithm) through the state space just as in our merging algorithm, only without the benefit of the value function bounds and the pruning of non-competitive actions.

## 4   Conclusion

The ability to perform multiple decision-making tasks simultaneously, and even to incorporate new tasks dynamically into ongoing previous tasks, is of obvious interest to both cognitive science and engineering. Using the framework of MDPs for individual decision-making tasks, we have reformulated the above problem as that of dynamically merging MDPs. We have presented a modified value iteration algorithm for dynamically merging MDPs, proved its convergence, and illustrated its use on a simple merging task.

As future work we intend to apply our merging algorithm to a real-world job-shop scheduling problem, extend the algorithm into the framework of semi-Markov decision processes, and explore the performance of the algorithm in the case where a model of the MDPs is not available.

a)

b)

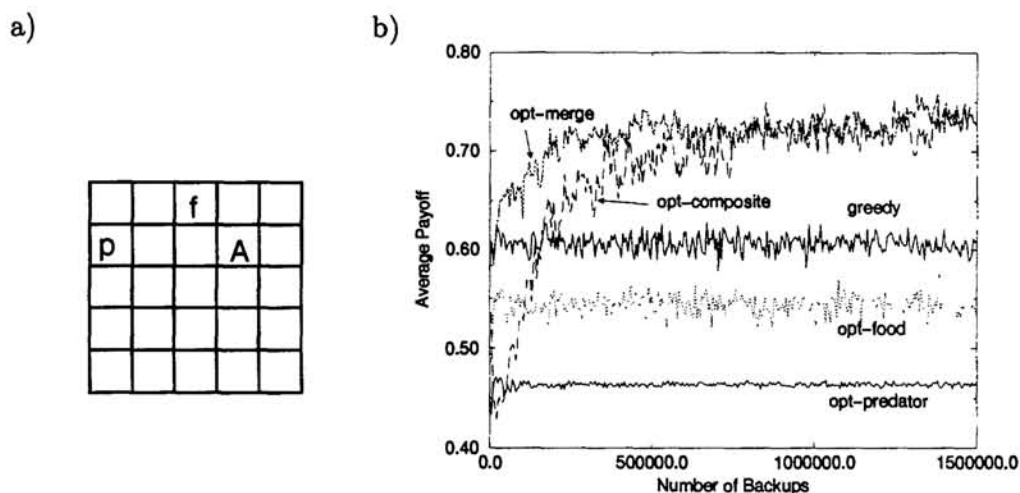

Figure 1: **a)** Our agent (A) roams an $n$ by $n$ grid. It gets a payoff of 0.5 for every time step it avoids predator (P), and earns a payoff of 1.0 for every piece of food (f) it finds. The agent moves two steps for every step P makes, and P always moves directly toward A. When food is found, it reappears at a random location on the next time step. On every time step, A has a 10% chance of ignoring its policy and making a random move. **b)** The mean payoff of different learning strategies vs. number of backups. The bottom two lines show that when trained on either task in isolation, a learner reaches the optimal payoff for that task in fewer than 5000 backups. The greedy approach makes no further backups, but performs well below optimal. The optimal composite solution, trained ab initio, requires requires nearly 1 million backups. Our algorithm begins with the 5000-backup solutions for the individual tasks, and converges to the optimum 4-5 times more quickly than the ab initio solution.

## Acknowledgements

Satinder Singh was supported by NSF grant IIS-9711753.

## Footnotes

[1]Recall that unsuperscripted quantities refer to the composite MDP while superscripted quantities refer to component MDPs. Also, $A$ is the set of actions that are available to the composite MDP after taking into account the constraints on picking actions simultaneously for the factor MDPs.

## References

Barto, A. G., Bradtke, S. J., & Singh, S. (1995). Learning to act using real-time dynamic programming. *Artificial Intelligence, 72*, 81–138.

Bertsekas, D. P. (1995). *Dynamic Programming and Optimal Control*. Belmont, MA: Athena Scientific.

Bertsekas, D. P. & Tsitsiklis, J. N. (1996). *Neuro-Dynamic Programming*. Belmont, MA: Athena Scientific.

Kaelbling, L. P. (1990). *Learning in Embedded Systems*. PhD thesis, Stanford University, Department of Computer Science, Stanford, CA. Technical Report TR-90-04.

Moore, A. W. & Atkeson, C. G. (1993). Prioritized sweeping: Reinforcement learning with less data and less real time. *Machine Learning, 13*(1).

Singh, S. (1997). Reinforcement learning in factorial environments. submitted.

Watkins, C. J. C. H. (1989). *Learning from Delayed Rewards*. PhD thesis, Cambridge Univ., Cambridge, England.

Zhang, W. & Dietterich, T. G. (1995). High-performance job-shop scheduling with a time delay TD(lambda) network. In *NIPSystems 8*. MIT Press.